# Bio-inspired Real Time Sensory Map Realignment in a Robotic Barn Owl

**Juan Huo, Zhijun Yang and Alan Murray**
DTC, School of Informatics, Schoolf of Electronics & Engineering
The University of Edinburgh
Edinburgh, UK
{J.Huo, Zhijun.Yang, Alan.Murray}@ed.ac.uk

## Abstract

The visual and auditory map alignment in the Superior Colliculus (SC) of barn owl is important for its accurate localization for prey behavior. Prism learning or Blindness may interfere this alignment and cause loss of the capability of accurate prey. However, juvenile barn owl could recover its sensory map alignment by shifting its auditory map. The adaptation of this map alignment is believed based on activity dependent axon developing in Inferior Colliculus (IC). A model is built to explore this mechanism. In this model, axon growing process is instructed by an inhibitory network in SC while the strength of the inhibition adjusted by Spike Timing Dependent Plasticity (STDP). We test and analyze this mechanism by application of the neural structures involved in spatial localization in a robotic system.

## 1 Introduction

Barn owl is a nocturnal predator with strong able auditory and visual localization system. During localization, the sensory stimuli are translated into neuron response, the visual and auditory maps are formed. In the deep Superior Colliculus (SC), visual and auditory information are integrated together. Normally, the object localization of visual map and auditory map are aligned with each other. But this alignment can be disrupted by wearing a prism or blindness [1, 2]. The juvenile barn owl is able to adapt so that it can foveates correctly on the source of auditory stimuli. A model based on the newest biological discoveries and account for a large amount of biological observations has been developed to explore the adaptation in map alignment [3].

This model is applied to a robotic system emulating the behavior of heading of the barn owl, so as to provide a real-time visual and auditory information integration and map realignment. It also provides a new mechanism for the hardware to mimic some of brain's abilities, adapt to novel situation without instruction.

### 1.1 Biological Background

Superior Colliculus (SC) gets different sensory inputs and it sends its outputs to effect behavior. As a hub of sensory information, SC neurons access the auditory stimuli from Inferior Colliculus (IC) [4, 1], which includes external Inferior Colliculus (ICx) and central Inferior Colliculus (ICc). ICx wraps around ICc. As revealed by anatomical and physiological experiments, the main site of map adaptation is in two areas, one is axon connection between ICc and ICx, the other area is an inhibitory network in SC.

Large amounts of evidence has shown axon sprouting and retraction between ICc and ICx are guided by inhibitory network in SC during prism learning [5, 6, 7]. Axons do not extend spontaneously,

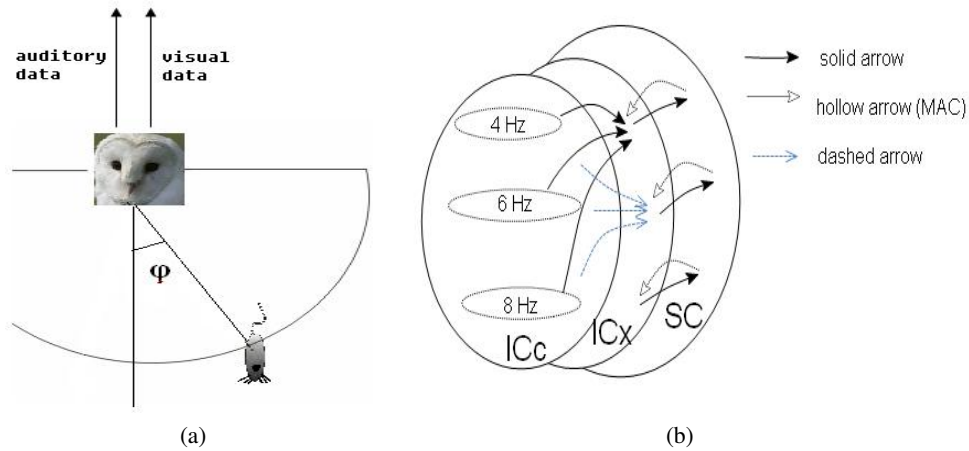

<div align="center">(a)            (b)</div>

Figure 1: (a) The simulation environment. (b) The information projection between ICc, ICx and SC.

they're promoted by neurotrophin (one kind of nerve growth factor) release and electrical activity of the cell body [8]. The release of neurotrophin is triggered by guiding signal from SC. In this paper we call the guiding signal, Map Adaptation Cue (MAC), as shown in Fig. 1(b). In the inhibitory network, MAC is assumed to be introduced by inter neuron, which is plausible to be bimodal neuron [7]. Bimodal neuron can be potentiated by both visual input (from retina) and auditory input (from ICx). Its neuron response is obviously strenthened when visual and auditory input are correlated [9]. Previous work has pointed out Hebbian Learning plays a main role in sensory information integration on bimodal neuron [4]. This paper includes a closer representation of biological structure.

## 2   Neural spike train

Neurons in nervous system process and transmit information by neural spikes. Sensory stimulus is coded by the spatiotemporal spike pattern before applied to the spiking neural network [10]. In this study, the input spike pattern was applied repeatedly and frequently, similar as the input stimuli. Spike patterns within which the fixed time intervals between spikes are set mannually, with two discrete values of mean firing rate, high and low. As the neuron response in visual map (retina layer) and auditory map (ICx layer) has a center surround profile, the receptive center has the highest firing rate, e.g. the "mexican hat", [11]. The spike patterns of visual center and auditory center, corresponding to a same target, are highly correlated with each other. Adjacent neurons respond with template spike trains of low firing rate. The spike patterns of center neuron and ajacent neuron are independent with each other. The remaining neurons have negligible activity. Another possilbe spike train generating method and its simulation result for this model can be found in paper [12].

## 3   Neural Model

The simulation is to emulate a virtual barn owl at the center of a fixed, head-centered reference system with the origin centered on the perch as in Fig. 1(a). Fig. 2(b) schematically illustrates the model, 4-layer (ICc, ICx, SC, retina), 10-pathway. Each pathway corresponds to $18°$ in azimuth. The single pathway is composed of two basic sections shown in Fig. 2(a). Block I comprises the ICc, ICx and the axon connections that map between them. Block II is both the detector of any shift between visual and auditory cues and the controller of the ICx/ICc mapping in block I. The connection between ICc and ICx in block I is instructed by Map Adaptation Cue (MAC), which is generated by the inter neuron in block II.

In block II, both bimodal and inter neurons in this model are Leaky Integrate-and-Fire (LIF) neuron (Equation 1). $g_e$ is the excitatory synaptic conductance, which is associated with excitatory reversal potential $V_{exc}$. Similarly, $g_i$, the inhibitory conductance, is associated with inhibitory reversal

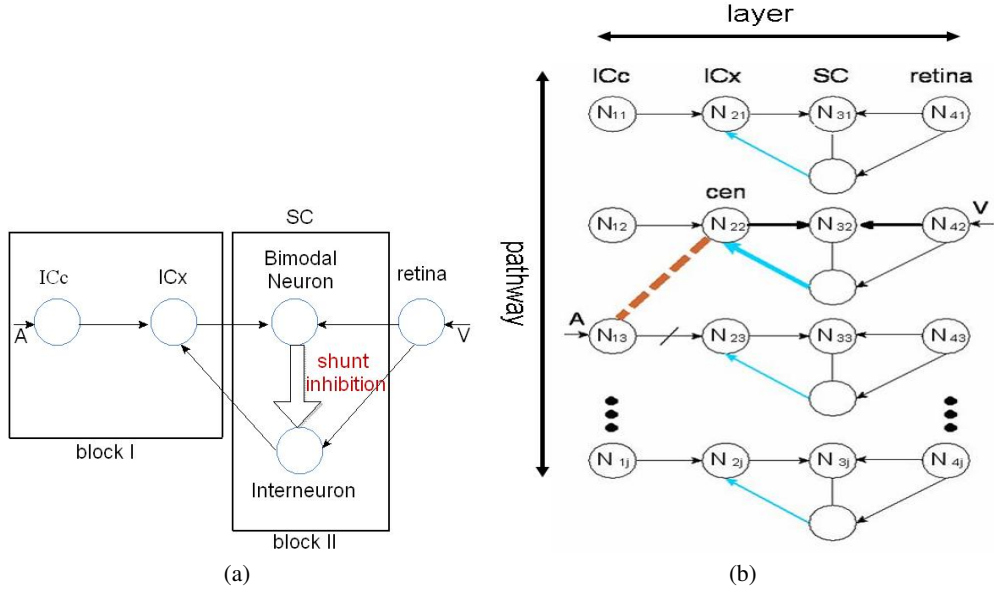

(a)                                    (b)

Figure 2: (a) Schematic of the auditory and visual signal processing pathway. (b) Schematic of the network. Each single pathway represents $18°$ in azimuth. The visual stimulus arrives in the retina at $N_{42}$, $N_{22}$ receives the strongest MAC. The active growthcone from $N_{13}$ is attracted by neurotrophin. The dashed line is the new connection built when the growthcone reaches its threshold. The old connection between $N_{13}$ and $N_{23}$ is thus eliminated due to lack of alignment between the auditory and visual stimuli.

potential $V_{inh}$. $g_l$ is the membrane conductance, the membrane resistance in this case is given by $R_m = 1/g_l$. When the membrane potential $V(t)$ reaches the threshold value of about -50 to -55$mV$, $V(t)$ is reset to a value $V_{reset}$ [13]. In this model, $V_{reset}$ is chosen to be equal to $V_{rest}$, the rest membrane potential, here $V_{rest} = V_{reset} = -70mV$. The other paprameters of the neuron model are as follows: $V_{exc} = 0mV$, $V_{inh} = -70mV$, $\tau_m = C_m R_m = 5ms$.

$$C_m \frac{dV(t)}{dt} = -g_l(V(t) - V_{rest}) - g_e(V(t) - V_{exc}) - g_i(V(t) - V_{inh}) \qquad (1)$$

The synapses connecting the sensory signals with the bimodel neuron are excitatory while the synapse between bimodal neuron and inter neuron is inhibitory. The synaptic weight change in this model is mediated by Spike Timing Dependent Plasticity (STDP). STDP is a learning rule in which the synaptic weight is strengthened or weakened by the paired presynaptic spikes and postsynaptic spikes in a time window [14].

The whole network is shown in Fig. 2(b), neuron $N_{ij}$ indicates the neuron location in layer i and pathway j. The developing of axon growthcone is activated by presynaptic spikes from its source layer ICc (layer 1). The target layer ICx (layer 2) releases neurotrophin when it is excited by MAC spikes. The concentration of neurotrophin $c_{2j}$ is set to be linearly proportional to the total MAC induced synaptic activity, $P_{2j}$, which sums the MAC spikes of ICx layer neurons. In Fig. 2(b), $N_{2j}(cen)$ is the ICx neuron that receives strongest stimulation from the visual signal, via the retina and SC. The concentrations of neurotrophin released by neurons $N_{2j}$ depend upon the distance between neuron $N_{2j}$ and $N_{2j}(cen)$, $\|N_{2j} - N_{2j}(cen)\|$. $c_{2j}$ is contributed by all active release sites, however, this contribution decays with distance. To represent the effect of neighbouring neurons, a spreading kernel $D(N_{2j} - N_{2j}(cen))$ is used to weight $P_{2j}$. $D(N_{2j} - N_{2j}(cen))$ is an exponential decay function with the decay variable $\|N_{2j} - N_{2j}(cen)\|$. The concentration of neurotrophin also decays with time step .

$$c(N_{2j}(cen)) \quad = \quad \sum_{N_{2j}} P(N_{2j})D(N_{2j} - N_{2j}(cen)) \tag{2}$$

$$= \quad \sum_{N_{2j}} P(N_{2j})e^{-\lambda\|N_{2j} - N_{2j}(cen)\|} \tag{3}$$

When there is neurotrophin released, the growth cone begins to grow induced by neural activity. The growth cone activity is bounded by the presynaptic factor which is a summation filter representing the linear sum of the presynaptic spikes of the corresponding neuron $N_{1j}$. The most active growth cone from source neuron $N_{1j}(sou)$ has the highest possibility to be extended. If $N_{2j}(tag)$ is the target direction of growth cone, $N_{2j}(tag)$ is identified when the accumulated neurotrophin $c_{2j}(tag)$ exceeds the threshold, the new connection between $N_{1j}(sou)$ and $N_{2j}(tag)$ is validated, meanwhile the neurotrophin is reset to the initial state. When the new connection is completed, the old connection which is bifurcated from the same neuron is blocked [15, 16].

$$N_{2j}(tag) = argmax_{N_{2j}(tag)\in Y(N_{2j})} c_{2j} \tag{4}$$

## 4 Real-time Learning and Adaptation

### 4.1 Experiments

To analyze the capability of the model in a real-time robotic system, an e-puck robot equipped with two lateral microphones, and a camera with a $30°$ prism is shown in Fig. 3. E-puck robot communicates with PC through bluetooth interface. We use e-puck robot to emulate barn owl head. The visual and auditory target (LED and loudspeaker) was fixed in one location and the owl-head robot moves into different directions manually or by motor command. The high firing rate spike pattern was fed into center neurons, which correspond to the target localization in space, in ICc or retina layer. In the network model, each pathway represents $18°$ field in space. We label the neurons corresponding to the azimuth angle $-90° \sim -72°$, pathway 1, so that azimuth angle $0° \sim 18°$ is represented by pathway 6.

The chirp from the loudspeaker is 1K Hz sine wave. The sound signal is processed by Fast Fourier Transform (FFT). When the average amplitude of the input signal above a threshold, the characteristic frequency $\bar{f}$ and phase $\Delta\phi$ between the left and right ear are calculated. With Equation 5 and 6, we get the interaural time difference $\Delta t$ and the target direction $\theta$ in azimuth. In this equation, V is sound speed, L is the diameter of the robot head.

$$\Delta t = \frac{\Delta\phi}{2\pi\bar{f}} \tag{5}$$

$$\theta = \frac{\Delta t V}{L} \tag{6}$$

### 4.2 Experiment Results

The experiment consisted of two steps: first, the owl-head robot without prism was positioned to head towards different directions in a random sequence. For every stimulation, a visual or an audio-visual target was presented at one of the 10 available locations. Secondly, the owl-head robot wearing prism with azimuth angle $36°$ was presented to randomly selected direction in azimuth. In each direction, the target stimuli repeated 75 times. Each stimuli introduce spike cluster in 40 time units. These spikes are binary signals with equal amplitude. Experiment results have shown that the system was able to adjust itself in different initial conditions.

The results of $0°$ target localization in the first experiment are shown in Fig. 4. Since visual and auditory signals are registered with each other, both the visual excitatory synapse (the arrow between $N_{4j}$ and $N_{3j}$ in Fig. 2(b)) and auditory excitatory synapse (the arrow between $N_{2j}$ and $N_{3j}$ in Fig. 2(b)) are strengthened. This means the bimodal neuron becomes more active. Because of

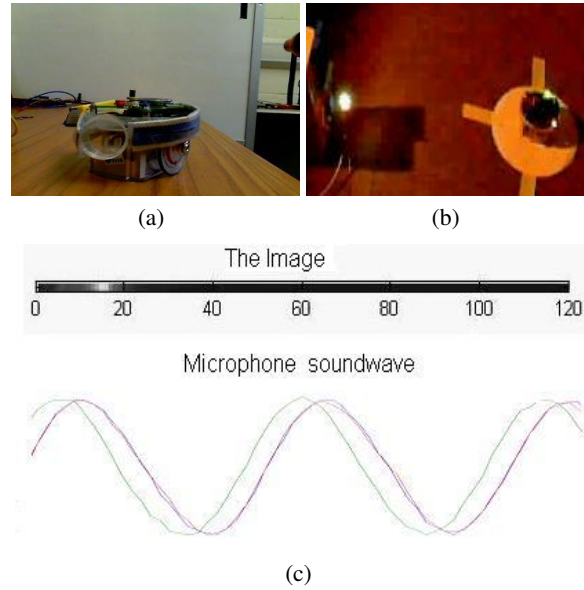

(a)　　　　　　　　　　(b)

(c)

Figure 3: (a) E-puck robot wearing a prism. (b) Real-time experiment. (c) Visual and auditory input. We get the visual direction from the luminous image by identifying the position of the brightest pixel. The auditory signal is processed by FFT to identify the phase difference between left and right ear, so as to find the auditory direction.

the inhibitory relationship between bimodal neuron and the interneuron, the interneuron is strongly inhibited and its output is close to zero. Therefore, no neurotrophin is released in the ICx neuron, as shown in Fig. 4(a). The growthcone does not grow in the auditory layer, so there is no change to the original axon connection, Fig. 4(b).

The results of $0°$ target localization in the second experiment are shown in Fig. 5 and Fig. 6. Because of the prism wearing, the visual receptive center and auditory receptive center are in different pathways, pathway 8 and pathway 6. Visual and auditory input spike trains are independent of each other in pathway 8. Thus both visual and auditory synapses connected to the bimodal neuron are weakened. The reduced inhibition increases the spike output of interneuron. This stimulates neurotrophin release in pathway 8. With high neurotrophin value and high firing rate spike train input, the pathway 6 growthcone is the most active one at the source layer. When the growthcone grows to certain level, the axon connection is updated, as shown in Fig. 6(b).

For the camera is limited by its visual angle $-30° \sim 30°$, the real-time robot experiment only tested pathway $4 \sim 7$. The rest of the pathway test is simulated in PC in terms of data accessed from pathway $4 \sim 7$. The last map realignment result is shown in Fig.7.

## 5  Conclusion

Adaptability is a crucial issue in the design of autonomous systems. In this paper, we demonstrate a robust model to eliminate the visual and auditory localization disparity. This model explains the mechanism behind visual and auditory signal integration. Spike-Timing Dependent Plasticity is accompanied by modulation of the signals between ICc and ICx neurons. The model also provides the first clear indication of the possible role of a "Map Adaptation Cue" in map alignment. The real-time application in a robotic barn owl head shows the model can work in real world which the brains of animals have to face.

By studying the brain wiring mechanism in superior colliculus, we can better understand the maps alignment in brain. Maps alignment also exists in hippacampus and cortex. It is believed maps alignment plays an important role for learning, perception and memory which is our future work.

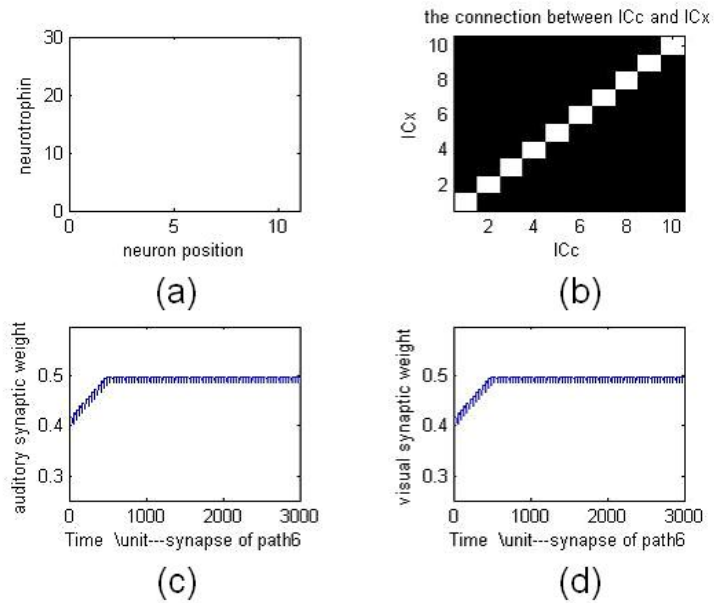

Figure 4: Visual and auditory localization signals from a same target are registered with each other. (a) There's no neurotrophin released from ICx layer at any time during the experiment. (b) The axon connection between ICc and ICx doesn't change. (c)(d) Here the target direction is in $0°$. Both visual and auditory receptive center corresponds to pathway 6 and their synaptic weight increases simultaneously.

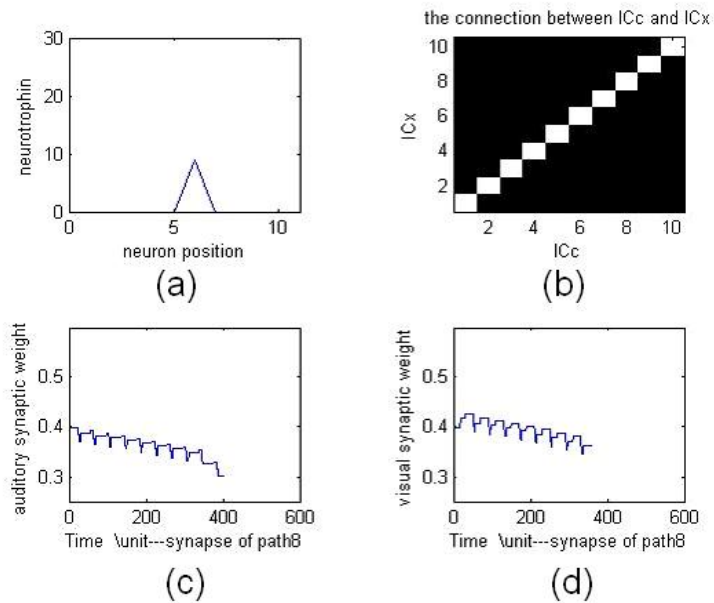

Figure 5: Visual and auditory localization signals are misaligned with each other. (a) Neurotrophin released from the target ICx neurons is accumulated. (b) The axon connection between ICc and ICx doesn't change before the neurontrophin and growthcone reach a threshold. Here the visual receptive center is in pathway 8, while the auditory receptive center is in pathway 6. (c)(d) Both the visual and auditory synapses are weakened because the input spike trains are independent with each other.

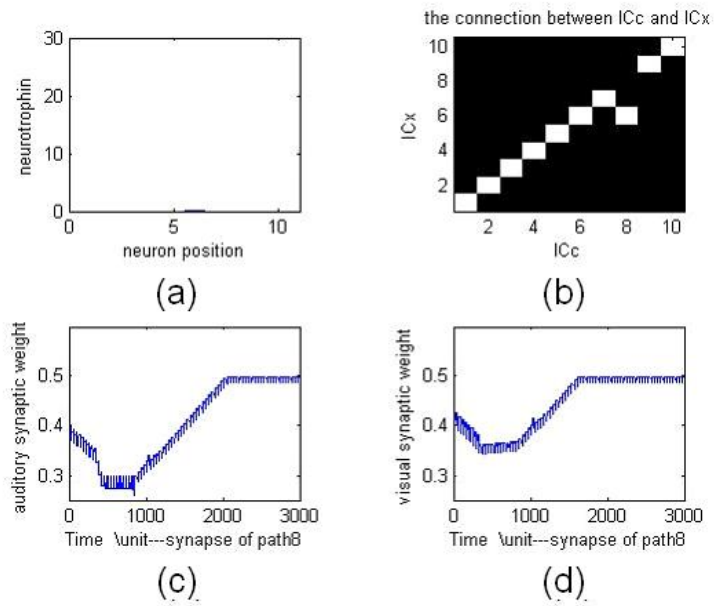

Figure 6: New axon connection is built. (a) After the axon connection is updated, neurotrophin is reset to its original status. (b) The new axon connection is built while the old connection is inhibited. (c)(d) Both visual and auditory synapses begin to increase after visual and auditory signal registered with each other again.

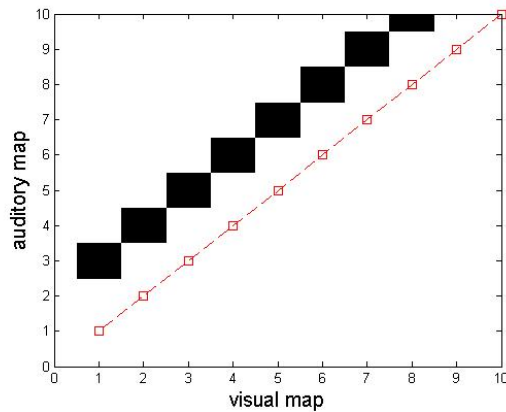

Figure 7: The arrangement of axon connection between maps. The small square represents the original point to point connection. The black blocks represent the new connection after adaptation.

Another issue for further discussion is MAC. Although it is clear MAC is generated by an inhibitory network in SC, whether it comes from bimodal neuron or not remains unclear.

The video of the experiment can be found on website:

http://www.see.ed.ac.uk/ s0454392/

**Acknowledgments**

For this research, we are grateful to Barbara Webb's suggestion for using e-puck robot. We would like to thank the support of the EPSRC Doctoral Training Center in Neuroinformatics. We also thank Leslie Smith for advice and assistance in model building.

# References

[1] E. Knudsen, "Auditory and visual maps of space in the optic tectum of the owl," *The Journal of Neuroscience*, vol. 2, pp. 1177–1194, 1982.

[2] K. EI, "Early blindness results in a degraded auditory map of space in the optic tectum of the barn owl," *Proc Natl Acad Sci*, vol. 85, no. 16, pp. 6211–4, 1988.

[3] J. Huo, A. Murray, L. Smith, and Z. Yang, "Adaptation of barn owl localization system with spike timing dependent plasticity." IEEE World Congress on Computational Intelligence, June 2008.

[4] M. Rucci, G. Tononi, and G. M. Edelman, "Registration of neural maps through value-dependent learning: modeling the alignment of auditory and visual maps in the barn owl's optic tectum," *J Neurosci.*, vol. 17, no. 1, pp. 334–52, 1997.

[5] J. I.Gold and K. EI, "Adaptive adjustment of connectivity in the inferior colliculus revealed by focal pharmacological inactivation," *J Neurophysiol.*, vol. 85(4), pp. 1575–84, 2001.

[6] P. S. Hyde and E. I. Knudsen, "Topographic projection from the optic tectum to the auditory space map in the inferior colliculus of the barn owl," *J Comp Neurol*, vol. 421, pp. 146–160, 2000.

[7] Y. Gutfreund, W. Zheng, and E. I. Knudsen, "Gated visual input to the central auditory system," *Science*, vol. 297, pp. 1556–1559, 2002.

[8] J. L. Goldberg, J. S. Espinosa, Y. Xu, N. Davidson, G. T. Kovacs, and B. A. Barres, "Retinal ganglion cells do not extend axons by default: Promotion by neurotrophic signaling and electrical activity," *Neuron*, vol. 33, pp. 689–702, 2002.

[9] M. Meredith, J. Nemitz, and B. Stein, "Determinants of multisensory integration in superior colliculus neurons. i. temporal factors." *J Neurosci.*, vol. 10, pp. 3215–29, 1987.

[10] R. Hosaka, O. Araki, and T. Ikeguchi, "Stdp provides the substrate for igniting synfire chains by spatiotemporal input patterns," *Neural Computation*, vol. 20, pp. 415–435, 2008.

[11] J. Sneyd, *Mathematical Modeling in Physiology, Cell Biology, and Immunology*. New Orleans, Louisiana: American Mathematical Society, January 2001, vol. 59.

[12] J. Huo, Z. Yang, and A. Murray, "Modeling visual and auditory integration of barn owl superior colliculus with stdp." IEEE CIS&RAM, June 2008.

[13] L.F.Abbott and P. Dayan, *Theoretical Neuroscience*. Cambridge: MIT Press, August 2001, ch. 6.

[14] S. Song, K. D. Miller, and L.F.Abbott, "Competitive hebbian learning through spike-timing-dependent synaptic plasticity," *Nature Neurosci*, vol. 3, pp. 919–926, 2000.

[15] E. W.Dent and F. B.Gertler, "Cytoskeletal dynamics and transport in growth cone mobility and axon guidance," *Neuron*, vol. 40, pp. 209–227, 2003.

[16] H. Hatt and D. O. Smith, "Synaptic depression related to presynaptic axon conduction block," *J. Physiol.*, vol. 259, no. 2, pp. 367–93, 1976.

